# The Asymptotic Convergence-Rate of Q-learning

**Cs. Szepesvári***

Research Group on Artificial Intelligence, "József Attila" University,
Szeged, Aradi vrt. tere 1, Hungary, H-6720
szepes@math.u-szeged.hu

## Abstract

In this paper we show that for discounted MDPs with discount factor $\gamma > 1/2$ the asymptotic rate of convergence of Q-learning is $O(1/t^{R(1-\gamma)})$ if $R(1-\gamma) < 1/2$ and $O(\sqrt{\log \log t / t})$ otherwise provided that the state-action pairs are sampled from a fixed probability distribution. Here $R = p_{\min}/p_{\max}$ is the ratio of the minimum and maximum state-action occupation frequencies. The results extend to convergent on-line learning provided that $p_{\min} > 0$, where $p_{\min}$ and $p_{\max}$ now become the minimum and maximum state-action occupation frequencies corresponding to the stationary distribution.

## 1 INTRODUCTION

Q-learning is a popular reinforcement learning (RL) algorithm whose convergence is well demonstrated in the literature (Jaakkola et al., 1994; Tsitsiklis, 1994; Littman and Szepesvári, 1996; Szepesvári and Littman, 1996). Our aim in this paper is to provide *an upper bound* for the convergence rate of (lookup-table based) Q-learning algorithms. Although, this upper bound is not strict, computer experiments (to be presented elsewhere) and the form of the lemma underlying the proof indicate that the obtained upper bound can be made strict by a slightly more complicated definition for $R$. Our results extend to learning on aggregated states (see (Singh et al., 1995)) and other related algorithms which admit a certain form of asynchronous stochastic approximation (see (Szepesvári and Littman, 1996)).

Present address: Associative Computing, Inc., Budapest, Konkoly Thege M. u. 29–33, HUNGARY-1121

## 2 Q-LEARNING

Watkins introduced the following algorithm to estimate the value of state-action pairs in discounted Markovian Decision Processes (MDPs) (Watkins, 1990):

$$Q_{t+1}(x,a) = (1 - \alpha_t(x,a))Q_t(x,a) + \alpha_t(x,a)(r_t(x,a) + \gamma \max_{b \in A} Q_t(y_t, b)). \quad (1)$$

Here $x \in X$ and $a \in A$ are states and actions, respectively, $X$ and $A$ are finite. It is assumed that some random sampling mechanism (e.g. simulation or interaction with a real Markovian environment) generates random samples of form $(x_t, a_t, y_t, r_t)$, where the probability of $y_t$ given $(x_t, a_t)$ is fixed and is denoted by $P(x_t, a_t, y_t)$, $E[r_t \mid x_t, a_t] = R(x, a)$ is the immediate average reward which is received when executing action $a$ from state $x$, $y_t$ and $r_t$ are assumed to be independent given the history of the learning-process, and also it is assumed that $\text{Var}[r_t \mid x_t, a_t] < C$ for some $C > 0$. The values $0 \le \alpha_t(x, a) \le 1$ are called the learning rate associated with the state-action pair $(x, a)$ at time $t$. This value is assumed to be zero if $(x, a) \ne (x_t, a_t)$, i.e. only the value of the actual state and action is reestimated in each step. If

$$\sum_{t=1}^{\infty} \alpha_t(x,a) = \infty \quad (2)$$

and

$$\sum_{t=1}^{\infty} \alpha_t^2(x,a) < \infty \quad (3)$$

then Q-learning is guaranteed to converge to the only fixed point $Q^*$ of the operator $T : \Re^{X \times A} \to \Re^{X \times A}$ defined by

$$(TQ)(x,a) = R(x,a) + \gamma \sum_{y \in X} P(x,a,y) \max_b Q(y,b)$$

(convergence proofs can be found in (Jaakkola et al., 1994; Tsitsiklis, 1994; Littman and Szepesvári, 1996; Szepesvári and Littman, 1996)). Once $Q^*$ is identified the learning agent can act optimally in the underlying MDP simply by choosing the action which maximizes $Q^*(x, a)$ when the agent is in state $x$ (Ross, 1970; Puterman, 1994).

## 3 THE MAIN RESULT

Condition (2) on the learning rate $\alpha_t(x, a)$ requires only that every state-action pair is visited infinitely often, which is a rather mild condition. In this article we take the stronger assumption that $\{(x_t, a_t)\}_t$ is a sequence of independent random variables with common underlying probability distribution. Although this assumption is not essential it simplifies the presentation of the proofs greatly. A relaxation will be discussed later. We further assume that the learning rates take the special form

$$\alpha_t(x,a) = \begin{cases} \frac{1}{S_t(x,a)}, & \text{if } (x,a) = (x_t, a_t); \\ 0, & \text{otherwise}, \end{cases}$$

where $S_t(x, a)$ is the number of times the state-action pair was visited by the process $(x_s, a_s)$ before time step $t$ plus one, i.e. $S_t(x, a) = 1 + \#\{(x_s, a_s) = (x, a), 1 \le s \le$

$t$}. This assumption could be relaxed too as it will be discussed later. For technical reasons we further assume that the absolute value of the random reinforcement signals $r_t$ admit a common upper bound. Our main result is the following:

THEOREM 3.1 *Under the above conditions the following relations hold asymptotically and with probability one:*

$$|Q_t(x,a) - Q^*(x,a)| \leq \frac{B}{t^{R(1-\gamma)}} \qquad (4)$$

*and*

$$|Q_t(x,a) - Q^*(x,a)| \leq B\sqrt{\frac{\log\log t}{t}}, \qquad (5)$$

*for some suitable constant $B > 0$. Here $R = p_{\min}/p_{\max}$, where $p_{\min} = \min_{(x,a)} p(x,a)$ and $p_{\max} = \max_{(x,a)} p(x,a)$, where $p(x,a)$ is the sampling probability of $(x,a)$.*

Note that if $\gamma \geq 1 - p_{\max}/2p_{\min}$ then (4) is the slower, while if $\gamma < 1 - p_{\max}/2p_{\min}$ then (5) is the slower. The proof will be presented in several steps.

**Step 1.** Just like in (Littman and Szepesvári, 1996) (see also the extended version (Szepesvári and Littman, 1996)) the main idea is to compare $Q_t$ with the simpler process

$$\hat{Q}_{t+1}(x,a) = (1 - \alpha_t(x,a))\hat{Q}_t(x,a) + \alpha_t(x,a)(r_t(x,a) + \gamma \max_b Q^*(y_t,b)). \qquad (6)$$

Note that the only (but rather essential) difference between the definition of $\hat{Q}_t$ and that of $Q_t$ is the appearance of $Q^*$ in the defining equation of $\hat{Q}_t$. Firstly, notice that as a consequence of this change the process $\hat{Q}_t$ clearly converges to $Q^*$ and this convergence may be investigated along each component $(x,a)$ separately using standard stochastic-approximation techniques (see e.g. (Wasan, 1969; Poljak and Tsypkin, 1973)).

Using simple devices one can show that the difference process $\Delta_t(x,a) = |Q_t(x,a) - \hat{Q}_t(x,a)|$ satisfies the following inequality:

$$\Delta_{t+1}(x,a) \leq (1 - \alpha_t(x,a))\Delta_t(x,a) + \gamma\alpha_t(x,a)(\|\Delta_t\| + \|\hat{Q}_t - Q^*\|). \qquad (7)$$

Here $\|\cdot\|$ stands for the maximum norm. That is the task of showing the convergence rate of $Q_t$ to $Q^*$ is reduced to that of showing the convergence rate of $\Delta_t$ to zero.

**Step 2.** We simplify the notation by introducing the abstract process whose update equation is

$$x_{t+1}(i) = \left(1 - \frac{1}{S_t(i)}\right) x_t(i) + \frac{\gamma}{S_t(i)}\left(\|x_t\| + \epsilon_t\right), \qquad (8)$$

where $i \in 1,2,\ldots,n$ can be identified with the state-action pairs, $x_t$ with $\Delta_t$, $\epsilon_t$ with $\hat{Q}_t - Q^*$, etc. We analyze this process in two steps. First we consider processes when the "perturbation-term" $\epsilon_t$ is missing. For such processes we have the following lemma:

LEMMA 3.2 *Assume that $\eta_1,\eta_2,\ldots,\eta_t,\ldots$ are independent random variables with a common underlying distribution $P(\eta_t = i) = p_i > 0$. Then the process $x_t$ defined*

*by*

$$x_{t+1}(i) = \begin{cases} \left(1 - \frac{1}{S_t(i)}\right) x_t(i) + \frac{\gamma}{S_t(i)} \|x_t\|, & \text{if } \eta_t = i; \\ x_t(i), & \text{if } \eta_t \neq i, \end{cases} \tag{9}$$

*satisfies*

$$\|x_t\| = O\left(\frac{1}{t^{R(1-\gamma)}}\right)$$

*with probability one (w.p.1), where* $R = \min_i p_i / \max_i p_i$.

*Proof.* (Outline) Let $T_0 = 0$ and

$$T_{k+1} = \min\{ t \geq T_k \mid \forall i = 1 \ldots n, \exists s = s(i) : \eta_s = i \},$$

i.e. $T_{k+1}$ is the smallest time after time $T_k$ such that during the time interval $[T_k + 1, T_{k+1}]$ all the components of $x_t(\cdot)$ are "updated" in Equation (9) at least once. Then

$$x_{T_{k+1}+1}(i) \leq \left(1 - \frac{1-\gamma}{S_k}\right) \|x_{T_k+1}\|, \tag{10}$$

where $S_k = \max_i S_k(i)$. This inequality holds because if $t_k(i)$ is the last time in $[T_k + 1, T_{k+1}]$ when the $i^{\text{th}}$ component is updated then

$$\begin{aligned} x_{T_{k+1}+1}(i) &= x_{t_k(i)+1}(i) = (1 - 1/S_{t_k(i)})x_{t_k(i)}(i) + \gamma/S_{t_k(i)}\|x_{t_k(i)}(\cdot)\| \\ &\leq (1 - 1/S_{t_k(i)})\|x_{t_k(i)}(\cdot)\| + \gamma/S_{t_k(i)}\|x_{t_k(i)}(\cdot)\| \\ &= \left(1 - \frac{1-\gamma}{S_{t_k(i)}}\right) \|x_{t_k(i)}(\cdot)\| \\ &\leq \left(1 - \frac{1-\gamma}{S_k}\right) \|x_{T_k+1}(\cdot)\|, \end{aligned}$$

where it was exploited that $\|x_t\|$ is decreasing. Now, iterating (10) backwards in time yields

$$x_{T_k+1}(\cdot) \leq \|x_0\| \prod_{j=0}^{k-1} \left(1 - \frac{1-\gamma}{S_j}\right).$$

Now, consider the following approximations: $T_k \approx Ck$, where $C \geq 1/p_{\min}$ ($C$ can be computed explicitly from $\{p_i\}$), $S_k \approx p_{\max}T_{k+1} \approx p_{\max}/p_{\min}(k+1) \approx (k+1)/R_0$, where $R_0 = 1/Cp_{\max}$. Then, using Large Deviation's Theory,

$$\prod_{j=0}^{k-1} \left(1 - \frac{1-\gamma}{S_j}\right) \approx \prod_{j=0}^{k-1} \left(1 - \frac{R_0(1-\gamma)}{j+1}\right) \approx \left(\frac{1}{k}\right)^{R_0(1-\gamma)} \tag{11}$$

holds w.p.1. Now, by defining $s = T_k + 1$ so that $s/C \approx k$ we get

$$\|x_s\| = \|x_{T_k+1}\| \leq \|x_0\| \left(\frac{1}{k}\right)^{R_0(1-\gamma)} \approx \|x_0\| \left(\frac{C}{s}\right)^{R_0(1-\gamma)} \leq \|x_0\| \left(\frac{C}{s}\right)^{R(1-\gamma)},$$

which holds due to the monotonicity of $x_t$ and $1/k^{R_0(1-\gamma)}$ and because $R = p_{\min}/p_{\max} \leq R_0$. $\qquad \square$

**Step 3.** Assume that $\gamma > 1/2$. Fortunately, we know by an extension of the Law of the Iterated Logarithm to stochastic approximation processes that the convergence

rate of $\|\hat{Q}_t - Q^*\|$ is $O\left(\sqrt{\log\log t/t}\right)$ (the uniform boundedness of the random reinforcement signals must be exploited in this step) (Major, 1973). Thus it is sufficient to provide a convergence rate estimate for the perturbed process, $x_t$, defined by (8), when $\epsilon_t = C\sqrt{\log\log t/t}$ for some $C > 0$. We state that the convergence rate of $\epsilon_t$ is faster than that of $x_t$. Define the process

$$z_{t+1}(i) = \begin{cases} \left(1 - \frac{1-\gamma}{S_t(i)}\right) z_t(i), & \text{if } \eta_t = i; \\ z_t(i), & \text{if } \eta_t \neq i. \end{cases} \tag{12}$$

This process clearly lower bounds the perturbed process, $x_t$. Obviously, the convergence rate of $z_t$ is $O(1/t^{1-\gamma})$ which is slower than the convergence rate of $\epsilon_t$ provided that $\gamma > 1/2$, proving that $\epsilon_t$ must be faster than $x_t$. Thus, asymptotically $\epsilon_t \leq (1/\gamma - 1)x_t$, and so $\|x_t\|$ is decreasing for large enough $t$. Then, by an argument similar to that of used in the derivation of (10), we get

$$x_{T_{k+1}+1}(i) \leq \left(1 - \frac{1-\gamma}{S_k}\right) \|x_{T_k+1}\| + \frac{\gamma}{s_k}\epsilon_{T_k}, \tag{13}$$

where $s_k = \min_i S_k(i)$. By some approximation arguments similar to that of Step 2, together with the bound $(1/n^\eta)\sum_s^n s^{\eta-3/2}\sqrt{\log\log s} \leq s^{-1/2}\sqrt{\log\log s}$, $1 > \eta > 0$, which follows from the mean-value theorem for integrals and the law of integration by parts, we get that $x_t \approx O(1/t^{R(1-\gamma)})$. The case when $\gamma \leq 1/2$ can be treated similarly.

**Step 5.** Putting the pieces together and applying them for $\Delta_t = \hat{Q}_t - Q_t$ yields Theorem 3.1.

## 4   DISCUSSION AND CONCLUSIONS

The most restrictive of our conditions is the assumption concerning the sampling of $(x_t, a_t)$. However, note that under a fixed learning policy the process $(x_t, a_t)$ is a (non-stationary) Markovian process and if the learning policy converges in the sense that $\lim_{t\to\infty} P(a_t \mid \mathcal{F}_t) = P(a_t \mid x_t)$ (here $\mathcal{F}_t$ stands for the history of the learning process) then the process $(x_t, a_t)$ becomes eventually stationary Markovian and the sampling distribution could be replaced by the stationary distribution of the underlying stationary Markovian process. If actions become asymptotically optimal during the course of learning then the support of this stationary process will exclude the state-action pairs whose action is sub-optimal, i.e. the conditions of Theorem 3.1 will no longer be satisfied. Notice that the proof of convergence of such processes still follows very similar lines to that of the proof presented here (see the forthcoming paper (Singh et al., 1997)), so we expect that the same convergence rates hold and can be proved using nearly identical techniques in this case as well.

A further step would be to find explicit expressions for the constant $B$ of Theorem 3.1. Clearly, $B$ depends heavily on the sampling of $(x_t, a_t)$, as well as the transition probabilities and rewards of the underlying MDP. Also the choice of harmonic learning rates is arbitrary. If a general sequence $\alpha_t$ were employed then the artificial "time" $T_t(x, a) = 1/\Pi_{j=0}^t (1 - \alpha_t(x, a))$ should be used (note that for the harmonic sequence $T_t(x, a) \approx t$). Note that although the developed bounds are asymptotic in their present forms, the proper usage of Large Deviation's Theory would enable us to develope non-asymptotic bounds.

Other possible ways to extend the results of this paper may include Q-learning when learning on aggregated states (Singh et al., 1995), Q-learning for alternating/simultaneous Markov games (Littman, 1994; Szepesvári and Littman, 1996) and any other algorithms whose corresponding difference process $\Delta_t$ satisfies an inequality similar to (7).

Yet another application of the convergence-rate estimate might be the convergence proof of some average reward reinforcement learning algorithms. The idea of those algorithms follows from a kind of Tauberian theorem, i.e. that discounted sums converge to the average value if the discount rate converges to one (see e.g. Lemma 1 of (Mahadevan, 1994; Mahadevan, 1996) or for a value-iteration scheme relying on this idea (Hordjik and Tijms, 1975)). Using the methods developed here the proof of convergence of the corresponding Q-learning algorithms seems quite possible. We would like to note here that related results were obtained by Bertsekas et al. et. al (see e.g. (Bertsekas and Tsitsiklis, 1996)).

Finally, note that as an application of this result we immediately get that the convergence rate of the model-based RL algorithm, where the transition probabilities and rewards are estimated by their respective averages, is clearly better than that of for Q-learning. Indeed, simple calculations show that the law of iterated logarithm holds for the learning process underlying model-based RL. Moreover, the exact expression for the convergence rate depends explicitly on how much computational effort we spend on obtaining the next estimate of the optimal value function, the more effort we spend the faster is the convergence. This bound thus provides a direct way to control the tradeoff between the computational effort and the convergence rate.

## Acknowledgements

This research was supported by OTKA Grant No. F20132 and by a grant provided by the Hungarian Educational Ministry under contract no. FKFP 1354/1997. I would like to thank András Krámli and Michael L. Littman for numerous helpful and thought-provoking discussions.

# References

Bertsekas, D. and Tsitsiklis, J. (1996). *Neuro-Dynamic Programming*. Athena Scientific, Belmont, MA.

Hordjik, A. and Tijms, H. (1975). A modified form of the iterative method of dynamic programming. *Annals of Statistics*, 3:203–208.

Jaakkola, T., Jordan, M., and Singh, S. (1994). On the convergence of stochastic iterative dynamic programming algorithms. *Neural Computation*, 6(6):1185–1201.

Littman, M. (1994). Markov games as a framework for multi-agent reinforcement learning. In *Proc. of the Eleventh International Conference on Machine Learning*, pages 157–163, San Francisco, CA. Morgan Kauffman.

Littman, M. and Szepesvári, C. (1996). A Generalized Reinforcement Learning Model: Convergence and applications. In *Int. Conf. on Machine Learning*. http://iserv.iki.kfki.hu/asl-publs.html.

Mahadevan, S. (1994). To discount or not to discount in reinforcement learning: A case study comparing R learning and Q learning. In *Proceedings of the Eleventh International Conference on Machine Learning*, pages 164–172, San Francisco, CA. Morgan Kaufmann.

Mahadevan, S. (1996). Average reward reinforcement learning: Foundations, algorithms, and empirical results. *Machine Learning*, 22(1,2,3):124–158.

Major, P. (1973). A law of the iterated logarithm for the Robbins-Monro method. *Studia Scientiarum Mathematicarum Hungarica*, 8:95–102.

Poljak, B. and Tsypkin, Y. (1973). Pseudogradient adaption and training algorithms. *Automation and Remote Control*, 12:83–94.

Puterman, M. L. (1994). *Markov Decision Processes — Discrete Stochastic Dynamic Programming*. John Wiley & Sons, Inc., New York, NY.

Ross, S. (1970). *Applied Probability Models with Optimization Applications*. Holden Day, San Francisco, California.

Singh, S., Jaakkola, T., and Jordan, M. (1995). Reinforcement learning with soft state aggregation. In *Proceedings of Neural Information Processing Systems*.

Singh, S., Jaakkola, T., Littman, M., and Csaba Szepesvá ri (1997). On the convergence of single-step on-policy reinforcement-learning al gorithms. *Machine Learning*. in preparation.

Szepesvári, C. and Littman, M. (1996). Generalized Markov Decision Processes: Dynamic programming and reinforcement learning algorithms. *Machine Learning*. in preparation, available as TR CS96-10, Brown Univ.

Tsitsiklis, J. (1994). Asynchronous stochastic approximation and q-learning. *Machine Learning*, 8(3–4):257–277.

Wasan, T. (1969). *Stochastic Approximation*. Cambridge University Press, London.

Watkins, C. (1990). *Learning from Delayed Rewards*. PhD thesis, King's College, Cambridge. QLEARNING.